# A COMPUTATIONALLY ROBUST ANATOMICAL MODEL FOR RETINAL DIRECTIONAL SELECTIVITY

Norberto M. Grzywacz
Center Biol. Inf. Processing
MIT, E25-201
Cambridge, MA 02139

Franklin R. Amthor
Dept. Psychol.
Univ. Alabama Birmingham
Birmingham, AL 35294

## ABSTRACT

We analyze a mathematical model for retinal directionally selective cells based on recent electrophysiological data, and show that its computation of motion direction is robust against noise and speed.

## INTRODUCTION

Directionally selective retinal ganglion cells discriminate direction of visual motion relatively independently of speed (Amthor and Grzywacz, 1988a) and with high contrast sensitivity (Grzywacz, Amthor, and Mistler, 1989). These cells respond well to motion in the "preferred" direction, but respond poorly to motion in the opposite, null, direction.

There is an increasing amount of experimental work devoted to these cells. Three findings are particularly relevant for this paper: 1- An inhibitory process asymmetric to every point of the receptive field underlies the directional selectivity of ON-OFF ganglion cells of the rabbit retina (Barlow and Levick, 1965). This distributed inhibition allows small motions anywhere in the receptive field center to elicit directionally selective responses. 2- The dendritic tree of directionally selective ganglion cells is highly branched and most of its dendrites are very fine (Amthor, Oyster and Takahashi, 1984; Amthor, Takahashi, and Oyster, 1988). 3- The distributions of excitatory and inhibitory synapses along these cells' dendritic tree appear to overlap. (Famiglietti, 1985).

Our own recent experiments elucidated some of the spatiotemporal properties of these cells' receptive field. In contrast to excitation, which is transient with stimulus, the inhibition is sustained and might arise from sustained amacrine cells (Amthor and Grzywacz, 1988a). Its spatial distribution is wide, extending to the borders of the receptive field center (Grzywacz and Amthor, 1988). Finally, the inhibition seems to be mediated by a high-gain shunting, not hyperpolarizing, synapse, that is, a synapse whose reversal potential is close to cell's resting potential (Amthor and Grzywacz, 1989).

In spite of this large amount of experimental work, theoretical efforts to put these pieces of evidence into a single framework have been virtually inexistent.

We propose a directional selectivity model based on our recent data on the inhibition's spatiotemporal and nonlinear properties. This model, which is an elaboration of the model of Torre and Poggio (1978), seems to account for several phenomena related to retinal directional selectivity.

## THE MODEL

Figure 1 illustrates the new model for retinal directional selectivity. In this model, a stimulus moving in the null direction progressively activates receptive field regions linked to synapses feeding progressively more distal dendrites of the ganglion cells. Every point in the receptive field center activates adjacent excitatory and inhibitory synapses. The inhibitory synapses are assumed to cause shunting inhibition. (We also formulated a pre-ganglionic version of this model, which however, is outside the scope of this paper).

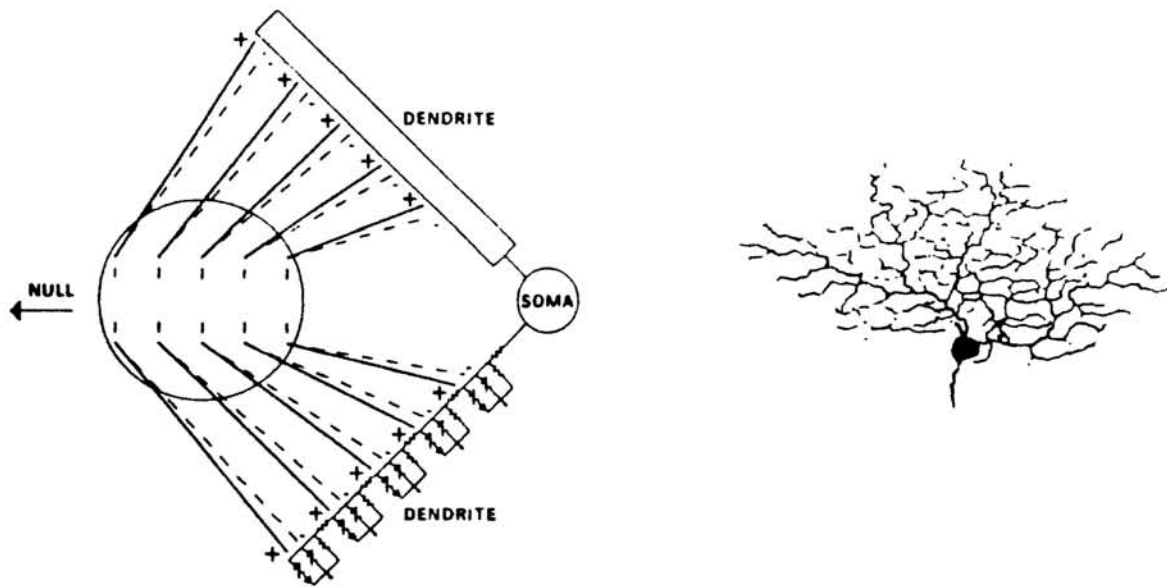

**FIGURE 1.** The new model for retinal directional selectivity.

This model is different than that of Poggio and Koch (1987), where the motion axis is represented as a sequence of activation of different dendrites. Furthermore, in their model, the inhibitory synapses must be closer to the soma than the excitatory ones. (However, our model is similar a model proposed, and argued against, elsewhere (Koch, Poggio, and Torre, 1982).

An advantage of our model is that it accounts for the large inhibitory areas to most points of the receptive field (Grzywacz and Amthor, 1988). Also, in the new model, the distributions of the excitatory and inhibitory synapses overlap along the dendritic tree, as suggested (Famiglietti, 1985). Finally, the dendritic tree of ON-OFF directionally selective ganglion cells (inset– Amthor, Oyster, and Takahashi,

1984) is consistent with our model. The tree's fine dendrites favor the multiplicity of directional selectivity and help to deal with noise (see below).

In this paper, we make calculations with an unidimensional version of the model dealing with motions in the preferred and null directions. Its receptive field maps into one dendrite of the cell. Set the origin of coordinates of the receptive field to be the point where a dot moving in the null direction enters the receptive field. Let $S$ be the size of the receptive field. Next, set the origin of coordinates in the dendrite to be the soma and let $\hat{L}$ be the length of the dendrite. The model postulates that a point $z$ in the receptive field activates excitatory and inhibitory synapses in point $x = z\hat{L}/S$ of the dendrite.

The voltages in the presynaptic sites are assumed to be linearly related to the stimulus, $I(z,t)$, that is, there are functions $f_e(t)$ and $f_i(t)$ such that the excitatory, $\beta_e(t)$, and inhibitory, $\beta_i(t)$, presynaptic voltages of the synapses to position $x$ in the dendrite are

$$\beta_j(x,t) = f_j(t) * I\left(\frac{xS}{L}, t\right), \qquad j = e, i,$$

where "$*$" stands for convolution. We assume that the integral of $f_e$ is zero, (the excitation is transient), and that the integral of $f_i$ is positive. (In practice, gamma distribution functions for $f_i$ and the derivatives of such functions for $f_e$ were used in this paper's simulations.)

The model postulates that the excitatory, $g_e$, and inhibitory, $g_i$, postsynaptic conductances are rectified functions of the presynaptic voltages. In some examples, we use the hyperbolic tangent as the rectification function:

$$g_j(x,t) = \frac{\gamma_j}{1 + e^{-(\beta_j(x,t) - T_j)}}, \qquad j = e, i,$$

where $\gamma_j$ and $T_j$ are constants. In other examples, we use the rectification functions described in Grzywacz and Koch (1987), and their model of ON–OFF rectifications.

For the postsynaptic site, we assume, without loss of generality, zero reversal potential and neglect the effect of capacitors (justified by the slow time–courses of excitation and inhibition).

Also, we assume that the inhibitory synapse leads to shunting inhibition, that is, its conductance is not in series with a battery. Let $E_e$ be the voltage of the excitatory battery. In general, the voltage, $V$, in different positions of the dendrite is described by the cable equation:

$$\frac{d^2 V(x,t)}{dx^2} = R_a \left(-E_e \tilde{g}_e(x,t) + V(x,t)\left(\tilde{g}_e(x,t) + \tilde{g}_i(x,t) + \tilde{g}_r\right)\right),$$

where $R_a$ is the axoplasm resistance per unit length, $g_r$ is the resting membrane conductance, and the tilde indicates that in this equation the conductances are given per unit length.

For the calculations illustrated in this paper, the stimuli are always delivered to the receptive field through two narrow slits. Thus, these stimuli activate synapses in two discrete positions of a dendrite. In this paper, we only show results for square wave and sinusoidal modulations of light, however, we also performed calculations for more general motions.

The synaptic sites are small so that their resting conductances are negligible, and we assume that outside these sites the excitatory and inhibitory conductances are zero. In this case, the equation for outside the synapses is:

$$\frac{d^2 U}{dy^2} = U,$$

where we defined $\lambda = 1/(R_a \tilde{g}_r)^{1/2}$ (the length constant), $U = V/E_e$, and $y = x/\lambda$. The boundary conditions used are

$$\frac{dU}{dy}\Big|_{y=L} = 0, \qquad \rho\frac{dU}{dy}\Big|_{y=0} = U(0),$$

where $L = \hat{L}/\lambda$, and where if $R_s$ is the soma's input resistance, then $\rho = R_s/(R_a\lambda)$ (the dendritic-to-soma conductance ratio). The first condition means that currents do not flow through the tips of the dendrites.

Finally, label by 1 the synapse proximal to the soma, and by 2 the distal one; the boundary conditions at the synapses are

$$\lim_{\substack{\nu \to \nu_j \\ \nu > \nu_j}} U = \lim_{\substack{\nu \to \nu_j \\ \nu < \nu_j}} U, \qquad j = 1, 2,$$

$$\frac{dU}{dy}\Big|_{\substack{\nu \to \nu_j \\ \nu > \nu_j}} = U\left(r_{e,j} + r_{i,j}\right) - r_{e,j} + \frac{dU}{dy}\Big|_{\substack{\nu \to \nu_j \\ \nu < \nu_j}}, \qquad j = 1, 2, \tag{1}$$

where $r_e = g_e R_a \lambda$ and $r_i = g_i R_a \lambda$.

It can be shown that the relative inhibitory strength for motions in the preferred direction decreases with $L$ and increases with $\rho$. Thus, to favor conditions for multiplicity of direction selectivity in the receptive field, we perform calculations with $L \to \infty$ and $\rho = 1$. The strengths of the excitatory synapses are set such that their contribution to somatic voltage in the absence of inhibition is invariant with position. Finally, we ensure that the excitatory synapses never saturate.

Under these conditions, one can show that the voltage in the soma is:

$$U(0) = \frac{(2r_{e,2} + (r_{e,2} + r_{i,2} + 2)\,r_{e,1})\,e^{2\delta y} - (r_{e,2} + r_{i,2})\,r_{e,1}}{((r_{i,1} + 2)\,r_{i,2} + 2r_{i,1} + 4)\,e^{2\delta y} - r_{i,1}r_{i,2}}, \tag{2}$$

where $\delta y$ is the distance between the synapses.

A final quantity, which is used in this paper is the directional selectivity index $\kappa$. Let $U_p$ and $U_n$ be the total responses to the second slit in the apparent motion in the preferred and null directions respectively. (Alternatively, for the sinusoidal

motion, these quantities are the respective average responses.) We follow Grzywacz and Koch (1987) and define

$$\kappa = \frac{U_p - U_n}{U_p + U_n}.$$ 

(3)

## RESULTS

This section presents the results of calculations based on the model. We address: the multiple computations of directional selectivity in the cells' receptive fields; the robustness of these computations against noise; the robustness of these computations against speed.

Figure 2 plots the degree of directional selectivity for apparent motions activating two synapses as function of the synapses' distance in a dendrite (computed from Equations 2 and 3).

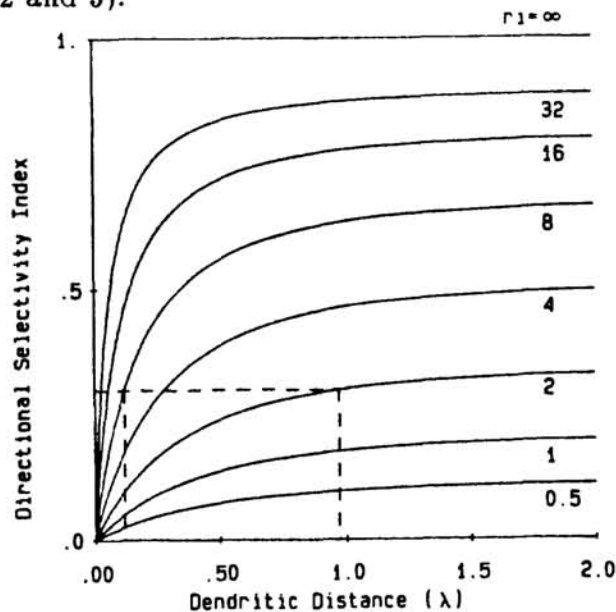

FIGURE 2. Locality of interaction between synapses activated by apparent motions.

It can be shown that the critical parameter controlling whether a certain synaptic distance produces a criterion directional selectivity is $r_i$ (Equation 1). As the parameter $r_i$ increases, the criterion distance decreases. Thus, since in retinal directionally selective cells the inhibition has high gain (Amthor and Grzywacz, 1989) and the dendrites are fine (Amthor, Oyster and Takahashi, 1984; Amthor, Takahashi, and Oyster, 1988), then $r_i$ is high, and motions anywhere in receptive field should elicit directionally selective responses (Barlow and Levick, 1965). In other words, the model's receptive field computes motion direction multiple times.

Next, we show that the high inhibitory gain and the cells' fine dendrites help to deal with noise, and thus, may explain the high contrast sensitivity (0.5% contrast-

Grzywacz, Amthor, and Mistler, 1989) of the cells' directional selectivity. This conclusion is illustrated in Figure 3's multiple plots.

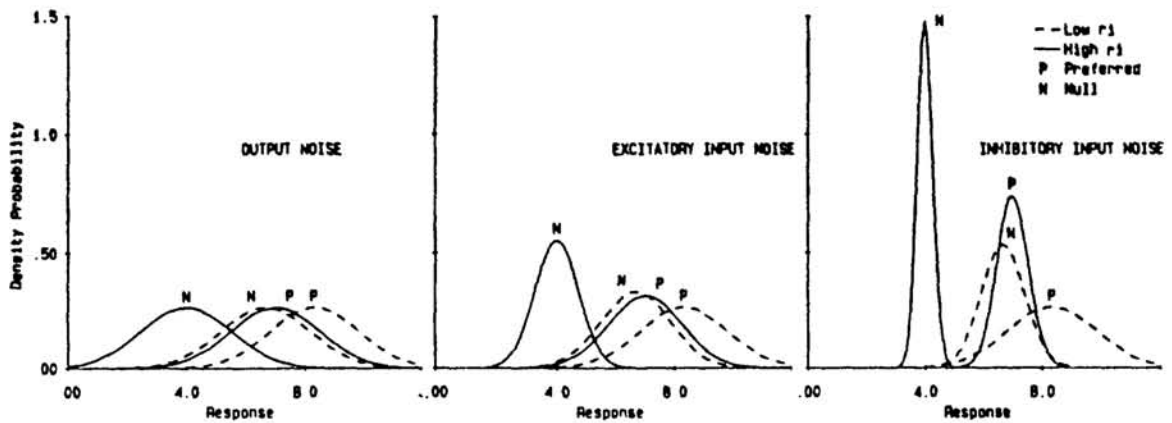

**FIGURE 3.** The model's computation of direction is robust against additive noise in the output, and in the excitatory and inhibitory inputs.

To generate this figure, we used Equation 2 assuming that a Gaussian noise is added to the cell's output, excitatory input, or inhibitory input. (In the latter case, we used an approximation that assumes small standard deviation for the inhibitory input's noise.)

Once again the critical parameter is the $r_i$ defined in Equation 1. The larger this parameter is, the better the model deals with noise. In the case of output noise, an increase of the parameter separates the preferred and null mean responses. For noise in the excitatory input, a parameter increase not only separates the means, but also reduces the standard deviation: Shunting inhibition SHUnTS down the noise. Finally, the most dramatic improvement occurs when the noise is in the inhibitory input. (In all these plots, the parameter increase is always by a factor of three.)

Since for retinal directionally selective ganglion cells, $r_i$ is high (high inhibitory gain and fine dendrites), we conclude that the cells' mechanism are particularly well suited to deal with noise.

For sinusoidal motions, the directional selectivity is robust for a large range of temporal frequencies provided that the frequencies are sufficiently low (Figure 4). (Nevertheless, the cell's preferred direction responses may be sharply tuned to either temporal frequency or speed– Amthor and Grzywacz, 1988).

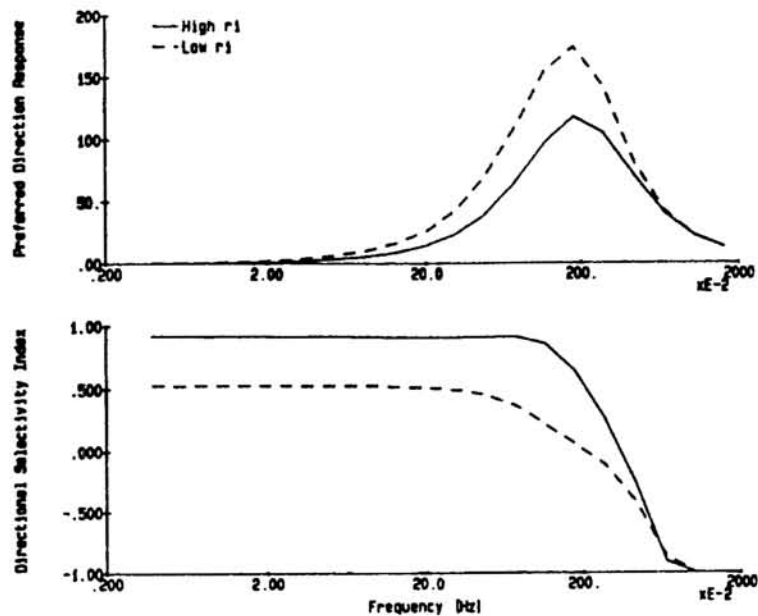

**FIGURE 4.** Directional selectivity is robust against speed modulation. To generate this curve, we subtracted the average response to a isolated flickering slit from the preferred and null average responses (from Equation 2).

This robustness is due to the invariance with speed for low speeds of the relative temporal phase shift between inhibition and excitation. Since the excitation has band-pass characteristics, it leads the stimulus by a constant phase. On the other hand, the inhibition is delayed and advanced in the preferred and null directions respectively, due to the asymmetric spatial integration. The phase shifts due to this integration are also speed invariant.

## CONCLUSIONS

We propose a new model for retinal directional selectivity. The shunting inhibition of ganglion cells (Torre and Poggio, 1978), which is temporally sustained, is the main biophysical mechanism of the model. It postulates that for null direction motion, the stimulus activates regions of the receptive field that are linked to excitatory and inhibitory synapses, which are progressively farther away from the soma. This models accounts for: 1- the distribution of inhibition around points of the receptive field (Grzywacz and Amthor, 1988); 2- the apparent full overlap of the distribution of excitatory and inhibitory synapses along the dendritic trees of directionally selective ganglion cells (Famiglietti, 1985); 3- the multiplicity of directionally selective regions (Barlow and Levick, 1965); 4- the high contrast sensitivity of the cells' directional selectivity (Grzywacz, Amthor, and Mistler, 1989); 5- the relative invariance of directional selectivity on stimulus speed (Amthor and Grzywacz, 1988).

Two lessons of our model to neural network modeling are: Threshold is not the only neural mechanism, and the basic computational unit may not be a neuron

but a piece of membrane (Grzywacz and Poggio, 1989). In our model, nonlinear interactions are relatively confined to specific dendritic tree branches (Torre and Poggio, 1978). This allows local computations by which *single* cells might generate receptive fields with multiple directionally selective regions, as observed by Barlow and Levick (1965). Such local computations could not occur if the inhibition only worked through a reduction in spike rate by somatic hyperpolarization.

Thus, most neural network models may be biologically irrelevant, since they are built upon a too simple model of the neuron. The properties of a network depend strongly on its basic elements. Therefore, to understand the computations of biological networks, it may be essential to first understand the basic biophysical mechanisms of information processing before developing complex networks.

## ACKNOWLEDGMENTS

We thank Lyle Borg–Graham and Tomaso Poggio for helpful discussions. Also, we thank Consuelita Correa for help with the figures. N.M.G. was supported by grant BNS-8809528 from the National Science Foundation, by the Sloan Foundation, and by a grant to Tomaso Poggio and Ellen Hildreth from the Office of Naval Research, Cognitive and Neural Systems Division. F.R.A. was supported by grants from the National Institute of Health (EY05070) and the Sloan Foundation.

## REFERENCES

Amthor & Grzywacz (1988) *Invest. Ophthalmol. Vis. Sci.* 29:225.

Amthor & Grzywacz (1989) Retinal Directional Selectivity Is Accounted for by Shunting Inhibition. Submitted for Publication.

Amthor, Oyster & Takahashi (1984) *Brain Res.* 298:187.

Amthor, Takahashi & Oyster (1989) *J. Comp. Neurol.* In Press.

Barlow & Levick (1965) *J. Physiol.* 178:477.

Famiglietti (1985) *Neurosci. Abst.* 11:337.

Grzywacz & Amthor (1988) *Neurosci. Abst.* 14:603.

Grzywacz, Amthor & Mistler (1989) Applicability of Quadratic and Threshold Models to Motion Discrimination in the Rabbit Retina. Submitted for Publication.

Grzywacz & Koch (1987) *Synapse* 1:417.

Grzywacz & Poggio (1989) In *An Introduction to Neural and Electronic Networks.* Zornetzer, Davis & Lau, Eds. Academic Press, Orlando, USA. In Press.

Koch, Poggio & Torre (1982) *Philos. Trans. R. Soc.* B 298:227.

Poggio & Koch (1987) *Sci. Am.* 256:46.

Torre & Poggio (1978) *Proc. R. Soc.* B 202:409.